# Modeling Saccadic Targeting in Visual Search

**Rajesh P. N. Rao**
Computer Science Department
University of Rochester
Rochester, NY 14627
rao@cs.rochester.edu

**Gregory J. Zelinsky**
Center for Visual Science
University of Rochester
Rochester, NY 14627
greg@cvs.rochester.edu

**Mary M. Hayhoe**
Center for Visual Science
University of Rochester
Rochester, NY 14627
mary@cvs.rochester.edu

**Dana H. Ballard**
Computer Science Department
University of Rochester
Rochester, NY 14627
dana@cs.rochester.edu

## Abstract

Visual cognition depends critically on the ability to make rapid eye movements known as *saccades* that orient the fovea over targets of interest in a visual scene. Saccades are known to be ballistic: the pattern of muscle activation for foveating a prespecified target location is computed prior to the movement and visual feedback is precluded. Despite these distinctive properties, there has been no general model of the saccadic targeting strategy employed by the human visual system during visual search in natural scenes. This paper proposes a model for saccadic targeting that uses iconic scene representations derived from oriented spatial filters at multiple scales. Visual search proceeds in a *coarse-to-fine* fashion with the largest scale filter responses being compared first. The model was empirically tested by comparing its performance with actual eye movement data from human subjects in a natural visual search task; preliminary results indicate substantial agreement between eye movements predicted by the model and those recorded from human subjects.

## 1  INTRODUCTION

Human vision relies extensively on the ability to make saccadic eye movements. These rapid eye movements, which are made at the rate of about three per second, orient the high-acuity foveal region of the eye over targets of interest in a visual scene. The high velocity of saccades, reaching up to 700° per second for large movements, serves to minimize the time in flight; most of the time is spent fixating the chosen targets.

The objective of saccades is currently best understood for reading text [13] where the eyes fixate almost every word, sometimes skipping over small function words. In general scenes, however, the purpose of saccades is much more difficult to analyze. It was originally suggested that

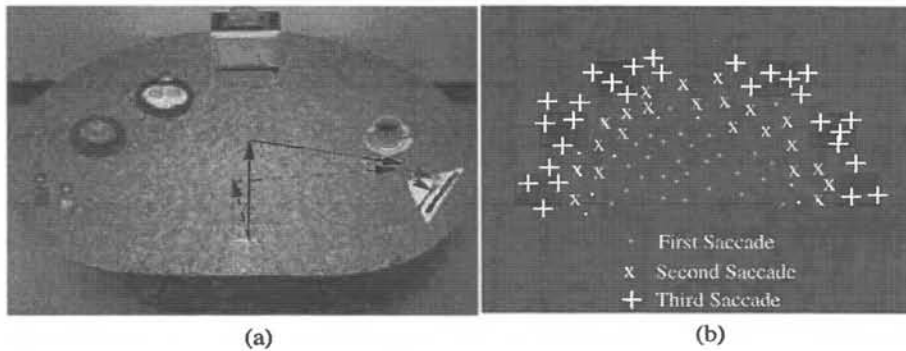

Figure 1: **Eye Movements in Visual Search**. (a) shows the typical pattern of multiple saccades (shown here for two different subjects) elicited during the course of searching for the object composed of the fork and knife. The initial fixation point is denoted by '+'. (b) depicts a summary of such movements over many experiments as a function of the six possible locations of a target object on the table.

the movements and their resultant fixations formed a visual-motor memory (or "scan-paths") of objects [11] but subsequent work has suggested that the role of saccades is more tightly coupled to the momentary problem solving strategy being employed by the subject. In chess, it has been shown that saccades are used to assess the current situation on the board in the course of making a decision to move, but the exact information that is being represented is not yet known [5]. In a task involving the copying of a model block pattern located on a board, fixations have been shown to be used in accessing crucial information for different stages of the copying task [2]. In natural language processing, there has been recent evidence that fixations reflect the instantaneous parsing of a spoken sentence [18]. However, none of the above work addresses the important question of what possible computational mechanisms underlie saccadic targeting.

The complexity of the targeting problem can be illustrated by the saccades employed by subjects to solve a natural visual search task. In this task, subjects are given a 1 second preview of a single object on a table and then instructed to determine, in the shortest possible amount of time, whether the previewed object is among a group of one to five objects on the same table in a subsequent view. The typical eye movements elicited are shown in Figure 1 (a). Rather than a single movement to the remembered target, several saccades are typical, with each successive saccade moving closer to the goal object (Figure 1 (b)).

The purpose of this paper is to describe a mechanism for programming saccades that can approximately model the saccadic targeting method used by human subjects. Previous models of human visual search have focused on simple search tasks involving elementary features such as horizontal/vertical bars of possibly different color [1, 4, 8] or have relied exclusively on bottom-up input-driven saliency criteria for generating scan-paths [10, 19]. The proposed model achieves targeting in arbitrary visual scenes by using bottom-up scene representations in conjunction with previously memorized top-down object representations; both of these representations are iconic, based on oriented spatial filters at multiple scales.

One of the difficult aspects of modeling saccadic targeting is that saccades are ballistic, i.e., their final location is computed prior to making the movement and the movement trajectory is uninterrupted by incoming visual signals. Furthermore, owing to the structure of the retina, the central 1.5° of the visual field is represented with a resolution that is almost 100 times greater than that of the periphery. We resolve these issues by positing that the targeting computation proceeds sequentially with coarse resolution information being used in the computation of target coordinates prior to fine resolution information. The method is compared to actual eye movements made by human subjects in the visual search task described above; the eye movements predicted by the model are shown to be in close agreement with observed human eye movements.

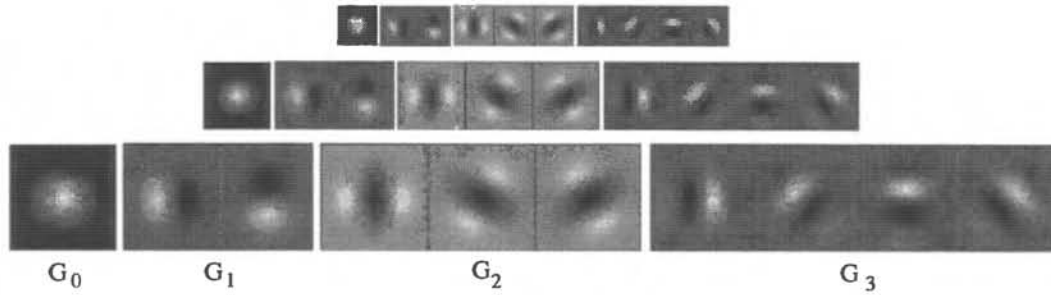

$G_0$        $G_1$              $G_2$                    $G_3$

Figure 2: **Multiscale Natural Basis Functions.** The 10 oriented spatial filters used in our model to generate iconic scene representations, shown here at three octave-separated scales. These filters resemble the receptive field profiles of cells in the primate visual cortex [20] and have been shown to approximate the dominant eigenvectors of natural image distributions as obtained from *principal component analysis* [7, 17].

## 2   ICONIC REPRESENTATIONS

The current implementation of our model uses a set of non-orthogonal basis functions as given by a zeroth order Gaussian $G_0$ and nine of its oriented derivatives as follows [6]:

$$G_n^{\theta_n}, n = 1, 2, 3, \theta_n = 0, \ldots, m\pi/(n+1), m = 1, \ldots, n \qquad (1)$$

where $n$ denotes the order of the filter and $\theta_n$ refers to the preferred orientation of the filter (Figure 2). The response of an image patch $I$ centered at $(x_0, y_0)$ to a particular basis filter $G_i^{\theta_j}$ can be obtained by convolving the image patch with the filter:

$$r_{i,j}(x_0, y_0) = \iint G_i^{\theta_j}(x_0 - x, y_0 - y)I(x, y)dx\,dy \qquad (2)$$

The iconic representation for the local image patch centered at $(x_0, y_0)$ is formed by combining into a high-dimensional vector the responses from the ten basis filters at different scales:

$$\mathbf{r_s}(x_0, y_0) = [r_{i,j,s}(x_0, y_0)] \qquad (3)$$

where $i = 0, 1, 2, 3$ denotes the order of the filter, $j = 1, \ldots, i+1$ denotes the different filters per order, and $s = s_{min}, \ldots, s_{max}$ denotes the different scales as given by the levels of a Gaussian image pyramid.

The use of multiple scales is crucial to the visual search model (see Section 3). In particular, the larger the number of scales, the greater the perspicuity of the representation as depicted in Figure 3. A multiscale representation also allows interpolation strategies for scale invariance. The high-dimensionality of the vectors makes them remarkably robust to noise due to the *orthogonality* inherent in high-dimensional spaces: given any vector, most of the other vectors in the space tend to be relatively uncorrelated with the given vector. The iconic representations can also be made invariant to rotations in the image plane (for a fixed scale) without additional convolutions by exploiting the property of *steerability* [6]. Rotations about an image plane axis are handled by storing feature vectors from different views. We refer the interested reader to [14] for more details regarding the above properties.

## 3   THE VISUAL SEARCH MODEL

Our model for visual search is derived from a model for vision that we previously proposed in [14]. This model decomposes visual behaviors into sequences of two visual routines, one for identifying the visual image near the fovea (the "what" routine), and another for locating a stored prototype on the retina (the "where" routine).

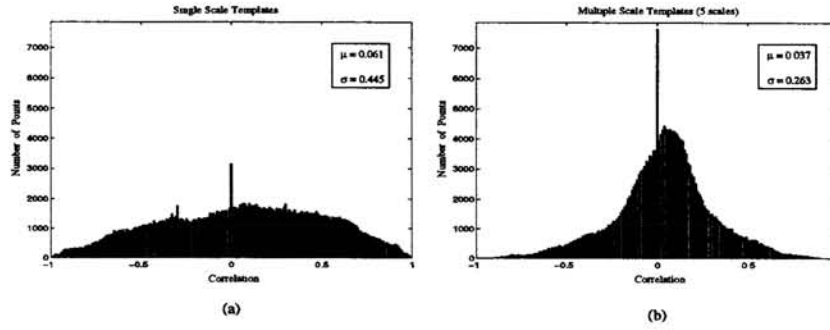

Figure 3: **The Effect of Scale.** The distribution of distances (in terms of correlations) between the response vector for a selected model point in the dining table scene and all other points in the scene is shown for single scale response vectors (a) and multiple scale vectors (b). Using responses from multiple scales (five in this case) results in greater perspicuity and a sharper peak near 0.0; only one point (the model point) had a correlation greater than 0.94 in the multiple scale case (b) whereas 936 candidate points fell in this category in the single scale case (a).

The visual search model assumes the existence of three independent processes running concurrently: (a) a *targeting process* (similar to the "where" routine of [14]) that computes the next location to be fixated; (b) an *oculomotor process* that accepts target locations and executes a saccade to foveate that location (see [16] for more details); and (c) a *decision process* that models the cortico-cortical dynamics of the $V1 \leftrightarrow V2 \leftrightarrow V4 \leftrightarrow IT$ pathway related to the identification of objects in the fovea (see [15] for more details).

Here, we focus on the saccadic targeting process. Objects of interest to the current search task are assumed to be represented by a *set* of previously memorized iconic feature vectors $\mathbf{r_s^m}$ where $s$ denotes the scale of the filters. The targeting algorithm computes the next location to be foveated as follows:

1. Initialize the routine by setting the current scale of analysis $k$ to the largest scale i.e. $k = max$; set $S_m(x, y) = 0$ for all $(x, y)$.

2. Compute the current *saliency image* $S_m$ as

$$S_m(x, y) = \sum_{s=k}^{max} \|\mathbf{r_s^i}(x, y) - \mathbf{r_s^m}\|^2 \qquad (4)$$

3. Find the location to be foveated by using the following *weighted population averaging (or soft max) scheme*:

$$(\hat{x}, \hat{y}) = \sum_{(x,y)} F(S_m(x, y))(x, y) \qquad (5)$$

where $F$ is an interpolation function. For the experiments, we chose:

$$F(S_m(x, y)) = \frac{e^{-S_m(x,y)/\lambda(k)}}{\sum_{(x,y)} e^{-S_m(x,y)/\lambda(k)}} \qquad (6)$$

This choice is attractive since it allows an interpretation of our algorithm as computing *maximum likelihood estimates* (cf. [12]) of target locations. In the above, $\lambda(k)$ is decreased with $k$.

4. Iterate step (2) and (3) above with $k = max\text{-}1, max\text{-}2, \ldots$ until either the target object has been foveated or the number of scales has been exhausted.

Figure 4 illustrates the above targeting procedure. The case where multiple model vectors are used per object proceeds in an analogous manner with the target location being averaged over all the model vectors.

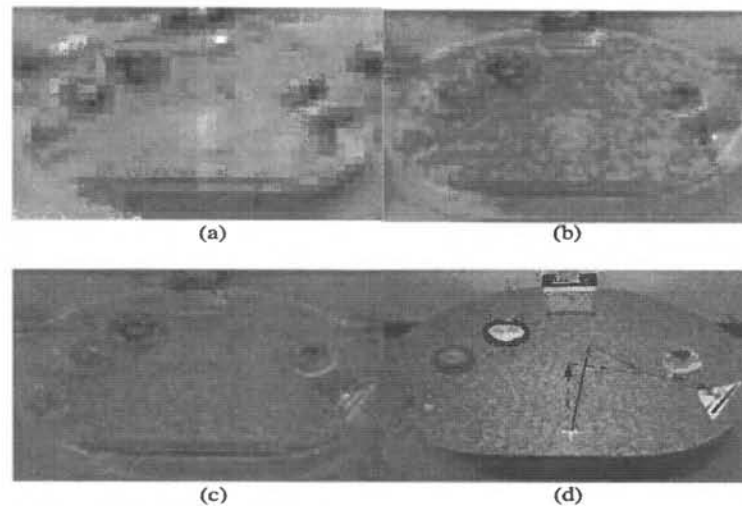

Figure 4: **Illustration of Saccadic Targeting.** The saliency image after the inclusion of the largest (a), intermediate (b), and smallest scale (c) as given by image distances to the prototype (the fork and knife); the lightest points are the closest matches. (d) shows the predicted eye movements as determined by the weighted population averaging scheme (for comparison, saccades from a human subject are given by the dotted arrows).

## 4   EXPERIMENTAL RESULTS AND DISCUSSION

Eye movements from four human subjects were recorded for the search task described in Section 1 for three different scenes (dining table, work bench, and a crib) using an *SRI Dual Purkinje* Eyetracker. The model was implemented on a pipeline image processor, the Datacube MV200, which can compute convolutions at frame rate $(30/sec)$. Figure 5 compares the model's performance to the human data. As the results show, there is remarkably good correspondence between the eye movements observed in human subjects and those generated by the model on the same data sets. The model has only one important parameter: the scaling function used to rate the peaks in the saliency map. In the development of the algorithm, this was adjusted to achieve an approximate fit to the human data.

Our model relies crucially on the existence of a coarse-to-fine matching mechanism. The main benefit of a coarse-to-fine strategy is that it allows continuous execution of the decision/oculomotor processes, thereby increasing the probability of an early match. Coarse-to-fine strategies have enjoyed recent popularity in computer vision with the advent of image pyramids in tasks such as motion detection [3]. Although these methods show that considerable speedup can be achieved by decreasing the size of window of analysis as resolution increases, our preliminary experiments suggest that this might be an inappropriate strategy for visual search: limiting search to a small window centered on the coarse location estimate obtained from a larger scale often resulted in significant errors since the targets frequently lay outside the search window. A possible solution is to adaptively select the size of the search window based on the current scene but this would require additional computational machinery.

A key question that remains is the source of sequential application of the filters in the human visual system. A possible source is the variation in resolution of the retina. Since only very high resolution information is at the fovea, and since this resolution falls off with distance, fine spatial scales may be ineffective purely because the fixation point is distant from the target. However, our preliminary experiments with modeling the variation in retinal resolution suggest that this is probably not the sole cause. The variations at middle distances from the fovea are too small to explain the dramatic improvement in target location experienced with the second saccade. Thus,

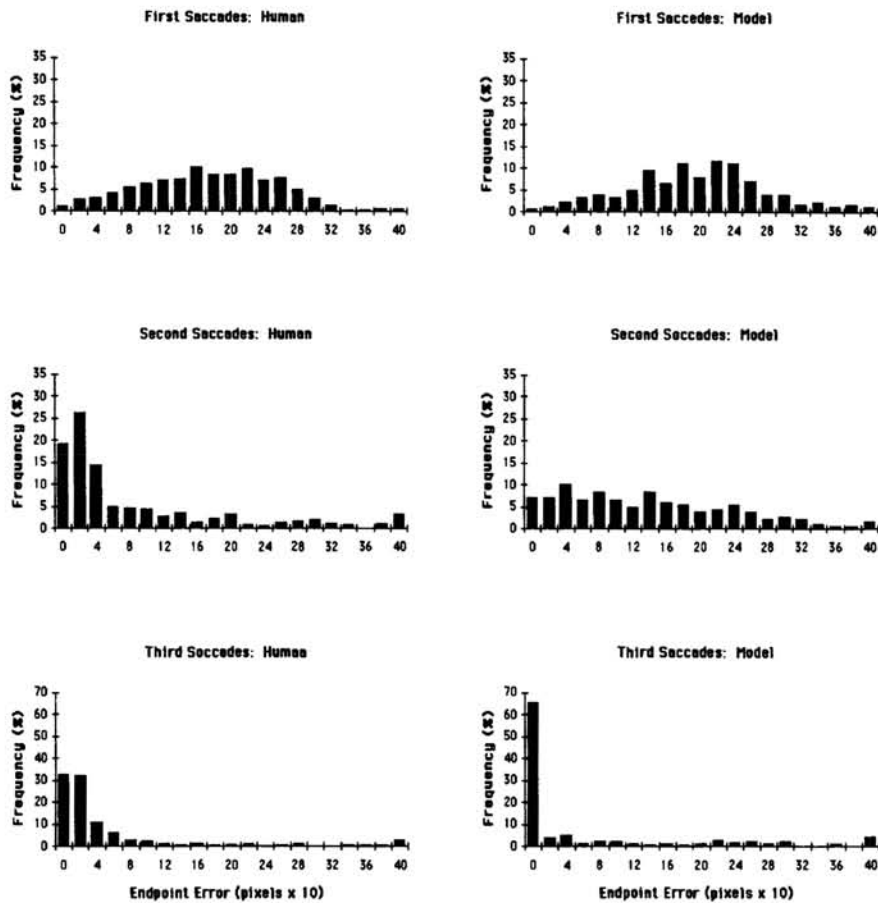

Figure 5: **Experimental Results.** The graphs compare the distribution of endpoint errors (in terms of frequency histograms) for three consecutive saccades as predicted by the model for 180 trials (on the right) and as observed with four human subjects for 676 trials (left). Each of the trials contained search scenes with one to five objects, one of the objects being the previewed model.

there are two remaining possibilities: (a) the resolution fall-off in the cortex is different from the retinal variation in a way that supports the data, or (b) the cortical machinery is set up to match the larger scales first. In the latter case, the observed data would result from the fact that the oculomotor system is ready to move before all the scales can be matched, and thus the eyes move to the current best target position. This interpretation of the data is appealing in two aspects. First, it reflects a long history of observations on the priority of large scale channels [9], and second, it reflects current thinking about eye movement programming suggesting that fixation times are approximately constant and that the eyes are moved as soon as they can be during the course of visual problem solving. The above questions can however be definitively answered only through additional testing of human subjects followed by subsequent modeling. We expect our saccadic targeting model to play a crucial role in this process.

## Acknowledgments

This research was supported by NIH/PHS research grants 1-P41-RR09283 and 1-R24-RR06853-02, and by NSF research grants IRI-9406481 and IRI-8903582.

# References

[1] Subutai Ahmad and Stephen Omohundro. Efficient visual search: A connectionist solution. In *Proceeding of the 13th Annual Conference of the Cognitive Science Society, Chicago*, 1991.

[2] Dana H. Ballard, Mary M. Hayhoe, and Polly K. Pook. Deictic codes for the embodiment of cognition. Technical Report 95.1, National Resource Laboratory for the study of Brain and Behavior, University of Rochester, January 1995.

[3] P.J. Burt. Attention mechanisms for vision in a dynamic world. In *ICPR*, pages 977–987, 1988.

[4] David Chapman. *Vision, Instruction, and Action*. PhD thesis, MIT Artificial Intelligence Laboratory, 1990. (Technical Report 1204).

[5] W.G. Chase and H.A. Simon. Perception in chess. *Cognitive Psychology*, 4:55–81, 1973.

[6] William T. Freeman and Edward H. Adelson. The design and use of steerable filters. *IEEE PAMI*, 13(9):891–906, September 1991.

[7] Peter J.B. Hancock, Roland J. Baddeley, and Leslie S. Smith. The principal components of natural images. *Network*, 3:61–70, 1992.

[8] Michael C. Mozer. *The perception of multiple objects : A connectionist approach*. Cambridge, MA: MIT Press, 1991.

[9] D. Navon. Forest before trees: The precedence of global features in visual perception. *Cognitive Psychology*, 9:353–383, 1977.

[10] Ernst Niebur and Christof Koch. Control of selective visual attention: Modeling the "where" pathway. This volume, 1996.

[11] D. Noton and L. Stark. Scanpaths in saccadic eye movements while viewing and recognizing patterns. *Vision Reseach*, 11:929–942, 1971.

[12] Steven J. Nowlan. Maximum likelihood competitive learning. In *Advances in Neural Information Processing Systems 2*, pages 574–582. Morgan Kaufmann, 1990.

[13] J.K. O'Regan. Eye movements and reading. In E. Kowler, editor, *Eye Movements and Their Role in Visual and Cognitive Processes*, pages 455–477. New York: Elsevier, 1990.

[14] Rajesh P.N. Rao and Dana H. Ballard. An active vision architecture based on iconic representations. *Artificial Intelligence (Special Issue on Vision)*, 78:461–505, 1995.

[15] Rajesh P.N. Rao and Dana H. Ballard. Dynamic model of visual memory predicts neural response properties in the visual cortex. Technical Report 95.4, National Resource Laboratory for the study of Brain and Behavior, Computer Sci. Dept., University of Rochester, November 1995.

[16] Rajesh P.N. Rao and Dana H. Ballard. Learning saccadic eye movements using multiscale spatial filters. In G. Tesauro, D.S. Touretzky, and T.K. Leen, editors, *Advances in Neural Information Processing Systems 7*, pages 893–900. Cambridge, MA: MIT Press, 1995.

[17] Rajesh P.N. Rao and Dana H. Ballard. Natural basis functions and topographic memory for face recognition. In *Proc. of IJCAI*, pages 10–17, 1995.

[18] M. Tanenhaus, M. Spivey-Knowlton, K. Eberhard, and J. Sedivy. Integration of visual and linguistic information in spoken language comprehension. To appear in Science, 1995.

[19] Keiji Yamada and Garrison W. Cottrell. A model of scan paths applied to face recognition. In *Proc. 17th Annual Conf. of the Cognitive Science Society*, 1995.

[20] R.A. Young. The Gaussian derivative theory of spatial vision: Analysis of cortical cell receptive field line-weighting profiles. *General Motors Research Publication GMR-4920*, 1985.